# TRAINING MULTILAYER PERCEPTRONS WITH THE EXTENDED KALMAN ALGORITHM

Sharad Singhal and Lance Wu
Bell Communications Research, Inc.
Morristown, NJ 07960

## ABSTRACT

A large fraction of recent work in artificial neural nets uses multilayer perceptrons trained with the back-propagation algorithm described by Rumelhart et. al. This algorithm converges slowly for large or complex problems such as speech recognition, where thousands of iterations may be needed for convergence even with small data sets. In this paper, we show that training multilayer perceptrons is an identification problem for a nonlinear dynamic system which can be solved using the Extended Kalman Algorithm. Although computationally complex, the Kalman algorithm usually converges in a few iterations. We describe the algorithm and compare it with back-propagation using two-dimensional examples.

## INTRODUCTION

Multilayer perceptrons are one of the most popular artificial neural net structures being used today. In most applications, the "back propagation" algorithm [Rumelhart et al, 1986] is used to train these networks. Although this algorithm works well for small nets or simple problems, convergence is poor if the problem becomes complex or the number of nodes in the network become large [Waibel et al, 1987]. In problems such as speech recognition, tens of thousands of iterations may be required for convergence even with relatively small data-sets. Thus there is much interest [Prager and Fallside, 1988; Irie and Miyake, 1988] in other "training algorithms" which can compute the parameters faster than back-propagation and/or can handle much more complex problems.

In this paper, we show that training multilayer perceptrons can be viewed as an identification problem for a nonlinear dynamic system. For linear dynamic

systems with white input and observation noise, the Kalman algorithm [Kalman, 1960] is known to be an optimum algorithm. Extended versions of the Kalman algorithm can be applied to nonlinear dynamic systems by linearizing the system around the current estimate of the parameters. Although computationally complex, this algorithm updates parameters consistent with all previously seen data and usually converges in a few iterations. In the following sections, we describe how this algorithm can be applied to multilayer perceptrons and compare its performance with back-propagation using some two-dimensional examples.

## THE EXTENDED KALMAN FILTER

In this section we briefly outline the Extended Kalman filter. Mathematical derivations for the Extended Kalman filter are widely available in the literature [Anderson and Moore, 1979; Gelb, 1974] and are beyond the scope of this paper.

Consider a nonlinear finite dimensional discrete time system of the form:

$$x(n+1) = f_n(x(n)) + g_n(x(n))w(n),$$
$$d(n) = h_n(x(n)) + v(n). \tag{1}$$

Here the vector $x(n)$ is the *state* of the system at time $n$, $w(n)$ is the *input*, $d(n)$ is the *observation*, $v(n)$ is observation noise and $f_n(\cdot)$, $g_n(\cdot)$, and $h_n(\cdot)$ are nonlinear vector functions of the state with the subscript denoting possible dependence on time. We assume that the initial state, $x(0)$, and the sequences $\{v(n)\}$ and $\{w(n)\}$ are independent and gaussian with

$$E[x(0)] = \bar{x}(0), \; E\{[x(0)-\bar{x}(0)][x(0)-\bar{x}(0)]'\} = P(0),$$
$$E[w(n)] = 0, \; E[w(n)w'(l)] = Q(n)\delta_{nl}, \tag{2}$$
$$E[v(n)] = 0, \; E[v(n)v'(l)] = R(n)\delta_{nl},$$

where $\delta_{nl}$ is the Kronecker delta. Our problem is to find an estimate $\hat{x}(n+1)$ of $x(n+1)$ given $d(j)$, $0 \leq j \leq n$. We denote this estimate by $\hat{x}(n+1|n)$.

If the nonlinearities in (1) are sufficiently smooth, we can expand them using Taylor series about the state estimates $\hat{x}(n|n)$ and $\hat{x}(n|n-1)$ to obtain

$$f_n(x(n)) = f_n(\hat{x}(n|n)) + F(n)[x(n)-\hat{x}(n|n)] + \cdots$$
$$g_n(x(n)) = g_n(\hat{x}(n|n)) + \cdots = G(n) + \cdots$$
$$h_n(x(n)) = h_n(\hat{x}(n|n-1)) + H'(n)[x(n)-\hat{x}(n|n-1)] + \cdots$$

where

$$G(n) = g_n(\hat{x}(n|n)),$$
$$F(n) = \frac{\partial f_n(x)}{\partial x}\bigg|_{x=\hat{x}(n|n)}, \; H'(n) = \frac{\partial h_n(x)}{\partial x}\bigg|_{x=\hat{x}(n|n-1)}. \tag{3}$$

i.e. $G(n)$ is the value of the function $g_n(\cdot)$ at $\hat{x}(n|n)$ and the $ij$th components of $F(n)$ and $H'(n)$ are the partial derivatives of the $i$th components of $f_n(\cdot)$ and $h_n(\cdot)$ respectively with respect to the $j$th component of $x(n)$ at the points indicated. Neglecting higher order terms and assuming

knowledge of $\hat{x}(n \mid n)$ and $\hat{x}(n \mid n-1)$, the system in (3) can be approximated as

$$x(n+1) = F(n)x(n) + G(n)w(n) + u(n) \quad n \geq 0 \tag{4}$$
$$z(n) = H'(n)x(n) + v(n) + y(n),$$

where

$$u(n) = f_n(\hat{x}(n \mid n)) - F(n)\hat{x}(n \mid n) \tag{5}$$
$$y(n) = h_n(\hat{x}(n \mid n-1)) - H'(n)\hat{x}(n \mid n-1).$$

It can be shown [Anderson and Moore, 1979] that the desired estimate $\hat{x}(n+1 \mid n)$ can be obtained by the recursion

$$\hat{x}(n+1 \mid n) = f_n(\hat{x}(n \mid n)) \tag{6}$$
$$\hat{x}(n \mid n) = \hat{x}(n \mid n-1) + K(n)[d(n) - h_n(\hat{x}(n \mid n-1))] \tag{7}$$
$$K(n) = P(n \mid n-1)H(n)[R(n) + H'(n)P(n \mid n-1)H(n)]^{-1} \tag{8}$$
$$P(n+1 \mid n) = F(n)P(n \mid n)F'(n) + G(n)Q(n)G'(n) \tag{9}$$
$$P(n \mid n) = P(n \mid n-1) - K(n)H'(n)P(n \mid n-1) \tag{10}$$

with $P(1 \mid 0) = P(0)$. $K(n)$ is known as the Kalman gain. In case of a *linear* system, it can be shown that $P(n)$ is the conditional error covariance matrix associated with the state and the estimate $\hat{x}(n+1 \mid n)$ is optimal in the sense that it approaches the conditional mean $E[x(n+1) \mid d(0) \cdots d(n)]$ for large $n$. However, for nonlinear systems, the filter is not optimal and the estimates can only loosely be termed conditional means.

## TRAINING MULTILAYER PERCEPTRONS

The network under consideration is a $L$ layer perceptron[1] with the $i$th input of the $k$th weight layer labeled as $z_i^{k-1}(n)$, the $j$th output being $z_j^k(n)$ and the weight connecting the $i$th input to the $j$th output being $\theta_{i,j}^k$. We assume that the net has $m$ inputs and $l$ outputs. Thresholds are implemented as weights connected from input nodes[2] with fixed unit strength inputs. Thus, if there are $N(k)$ nodes in the $k$th node layer, the total number of weights in the system is

$$M = \sum_{k=1}^{L} N(k-1)[N(k)-1]. \tag{11}$$

Although the inputs and outputs are dependent on time $n$, for notational brevity, we will not show this dependence unless explicitly needed.

---

1. We use the convention that the number of layers is equal to the number of weight layers. Thus we have $L$ layers of *weights* labeled $1 \cdots L$ and $L+1$ layers of *nodes* (including the input and output nodes) labeled $0 \cdots L$. We will refer to the $k$th weight layer or the $k$th node layer unless the context is clear.

2. We adopt the convention that the 1st input node is the threshold. i.e. $\theta_{1,j}^k$ is the threshold for the $j$th output node from the $k$th weight layer.

In order to cast the problem in a form for recursive estimation, we let the weights in the network constitute the state $x$ of the nonlinear system, i.e.

$$x = [\theta^L_{1,2}, \theta^L_{1,3} \cdots \theta^1_{N(0),N(1)}]^t. \tag{12}$$

The vector $x$ thus consists of all weights arranged in a linear array with dimension equal to the total number of weights $M$ in the system. The system model thus is

$$x(n+1) = x(n) \quad n > 0, \tag{13}$$
$$d(n) = z^L(n) + v(n) = h_n(x(n), z^0(n)) + v(n), \tag{14}$$

where at time $n$, $z^0(n)$ is the input vector from the training set, $d(n)$ is the corresponding desired output vector, and $z^L(n)$ is the output vector produced by the net. The components of $h_n(\cdot)$ define the nonlinear relationships between the inputs, weights and outputs of the net. If $\Gamma(\cdot)$ is the nonlinearity used, then $z^L(n) = h_n(x(n), z^0(n))$ is given by

$$z^L(n) = \Gamma\{(\theta^L)^t \Gamma\{(\theta^{L-1})^t \Gamma \cdots \Gamma\{(\theta^1)^t z^0(n)\} \cdots \}\}. \tag{15}$$

where $\Gamma$ applies componentwise to vector arguments. Note that the input vectors appear only implicitly through the observation function $h_n(\cdot)$ in (14). The initial state (before training) $x(0)$ of the network is defined by populating the net with gaussian random variables with a $N(\bar{x}(0), P(0))$ distribution where $\bar{x}(0)$ and $P(0)$ reflect any apriori knowledge about the weights. In the absence of any such knowledge, a $N(0, 1/\epsilon\, I)$ distribution can be used, where $\epsilon$ is a small number and $I$ is the identity matrix. For the system in (13) and (14), the extended Kalman filter recursion simplifies to

$$\hat{x}(n+1) = \hat{x}(n) + K(n)[d(n) - h_n(\hat{x}(n), z^0(n))] \tag{16}$$
$$K(n) = P(n)H(n)[R(n) + H^t(n)P(n)H(n)]^{-1} \tag{17}$$
$$P(n+1) = P(n) - K(n)H^t(n)P(n) \tag{18}$$

where $P(n)$ is the (approximate) conditional error covariance matrix.

Note that (16) is similar to the weight update equation in back-propagation with the last term $[z^L - h_n(\hat{x}, z^0)]$ being the error at the output layer. However, unlike the delta rule used in back-propagation, this error is propagated to the weights through the Kalman gain $K(n)$ which updates each weight through the entire gradient matrix $H(n)$ and the conditional error covariance matrix $P(n)$. In this sense, the Kalman algorithm is not a local training algorithm. However, the inversion required in (17) has dimension equal to the number of outputs $l$, not the number of weights $M$, and thus does not grow as weights are added to the problem.

## EXAMPLES AND RESULTS

To evaluate the output and the convergence properties of the extended Kalman algorithm, we constructed mappings using two-dimensional inputs with two or four outputs as shown in Fig. 1. Limiting the input vector to 2 dimensions allows us to visualize the decision regions obtained by the net and

to examine the outputs of any node in the net in a meaningful way. The x- and y-axes in Fig. 1 represent the two inputs, with the origin located at the center of the figures. The numbers in the figures represent the different output classes.

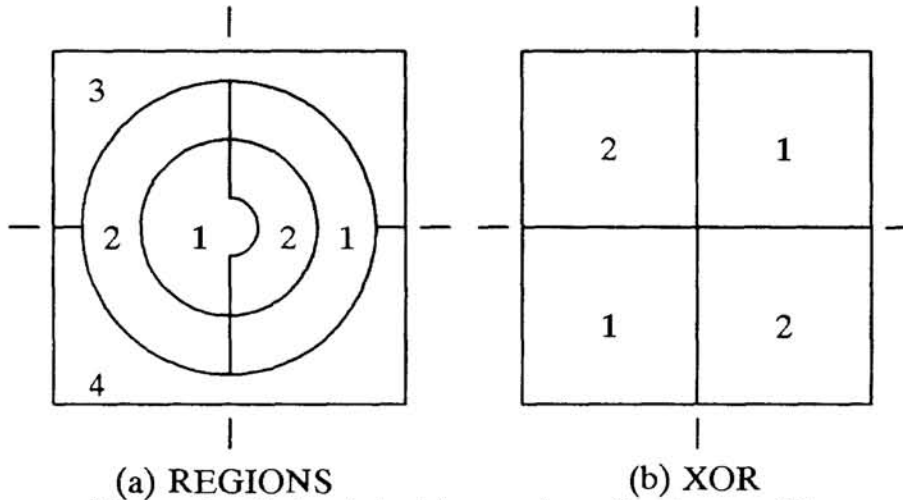

(a) REGIONS          (b) XOR
**Figure 1.** Output decision regions for two problems

The training set for each example consisted of 1000 random vectors uniformly filling the region. The hyperbolic tangent nonlinearity was used as the nonlinear element in the networks. The output corresponding to a class was set to 0.9 when the input vector belonged to that class, and to -0.9 otherwise. During training, the weights were adjusted after each data vector was presented. Up to 2000 sweeps through the input data were used with the stopping criteria described below to examine the convergence properties. The order in which data vectors were presented was randomized for each sweep through the data. In case of back-propagation, a convergence constant of 0.1 was used with no "momentum" factor. In the Kalman algorithm $R$ was set to $I \cdot e^{-k/50}$, where $k$ was the iteration number through the data. Within each iteration, $R$ was held constant.

**The Stopping Criteria**

Training was considered complete if any one of the following conditions was satisfied:

a.  2000 sweeps through the input data were used,

b.  the RMS (root mean squared) error at the output averaged over all training data during a sweep fell below a threshold $t_1$, or

c.  the error reduction $\delta$ after the $i$th sweep through the data fell below a threshold $t_2$, where $\delta_i = \beta\delta_{i-1} + (1-\beta)|e_i - e_{i-1}|$. Here $\beta$ is some positive constant less than unity, and $e_i$ is the error defined in b.

In our simulations we set $\beta = 0.97$, $t_1 = 10^{-2}$ and $t_2 = 10^{-5}$.

**Example 1 - Meshed, Disconnected Regions:**

Figure 1(a) shows the mapping with 2 disconnected, meshed regions surrounded by two regions that fill up the space. We used 3-layer perceptrons with 10 hidden nodes in each hidden layer to Figure 2 shows the RMS error obtained during training for the Kalman algorithm and back-propagation averaged over 10 different initial conditions. The number of sweeps through the data (x-axis) are plotted on a logarithmic scale to highlight the initial reduction for the Kalman algorithm. Typical solutions obtained by the algorithms at termination are shown in Fig. 3. It can be seen that the Kalman algorithm converges in fewer iterations than back-propagation and obtains better solutions.

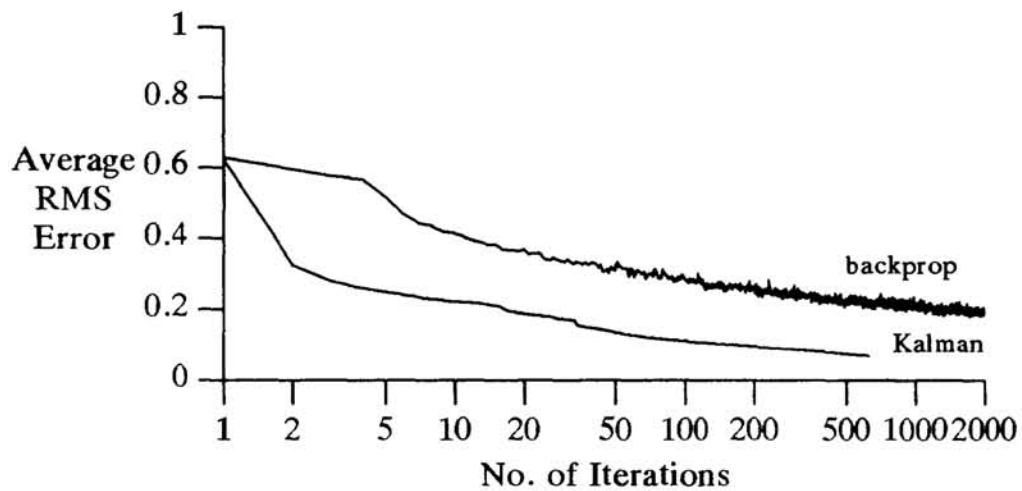

**Figure 2.** Average output error during training for Regions problem using the Kalman algorithm and backprop

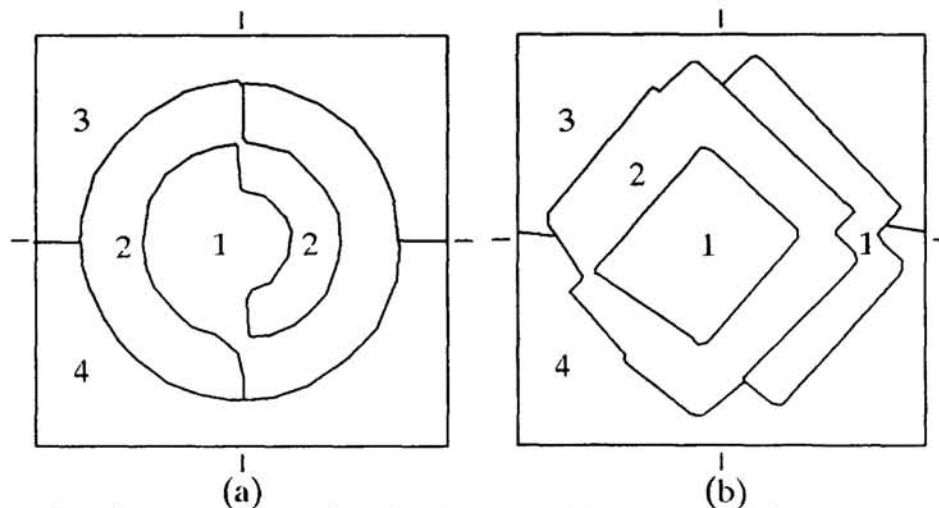

**Figure 3.** Typical solutions for Regions problem using (a) Kalman algorithm and (b) backprop.

**Example 2 - 2 Input XOR:**

Figure 1(b) shows a generalized 2-input XOR with the first and third quadrants forming region 1 and the second and fourth quadrants forming region 2. We attempted the problem with two layer networks containing 2-4 nodes in the hidden layer. Figure 4 shows the results of training averaged over 10 different randomly chosen initial conditions. As the number of nodes in the hidden layer is increased, the net converges to smaller error values. When we examine the output decision regions, we found that none of the nets attempted with back-propagation reached the desired solution. The Kalman algorithm was also unable to find the desired solution with 2 hidden nodes in the network. However, it reached the desired solution with 6 out of 10 initial conditions with 3 hidden nodes in the network and 9 out of 10 initial conditions with 4 hidden nodes. Typical solutions reached by the two algorithms are shown in Fig. 5. In all cases, the Kalman algorithm converged in fewer iterations and in all but one case, the final average output error was smaller with the Kalman algorithm.

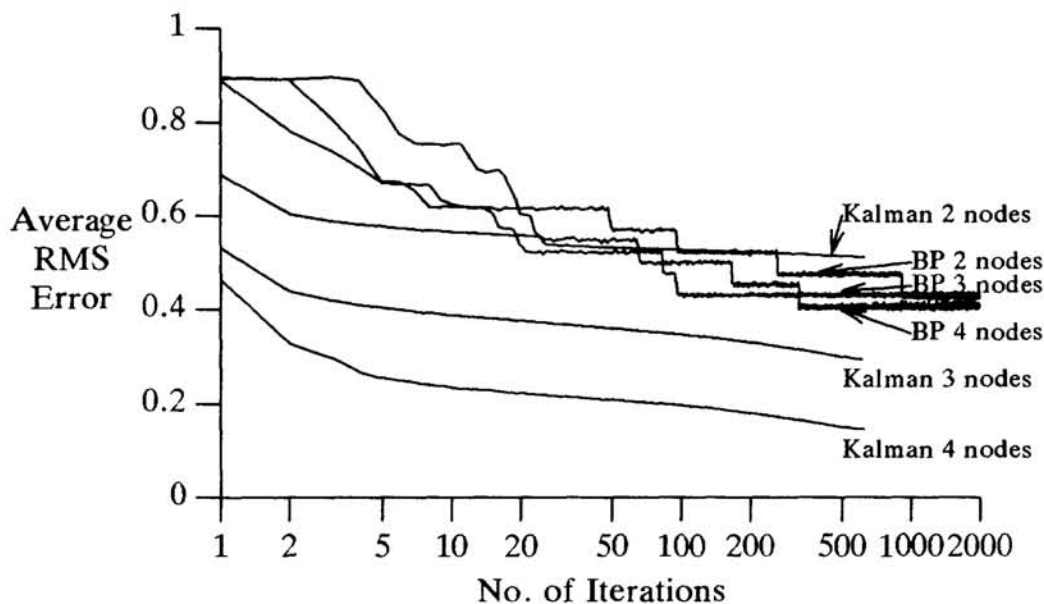

**Figure 4.** Average output error during training for XOR problem using the Kalman algorithm and backprop

## CONCLUSIONS

In this paper, we showed that training feed-forward nets can be viewed as a system identification problem for a nonlinear dynamic system. For linear dynamic systems, the Kalman filter is known to produce an optimal estimator. Extended versions of the Kalman algorithm can be used to train feed-forward networks. We examined the performance of the Kalman algorithm using artificially constructed examples with two inputs and found that the algorithm typically converges in a few iterations. We also used back-propagation on the same examples and found that invariably, the Kalman algorithm converged in

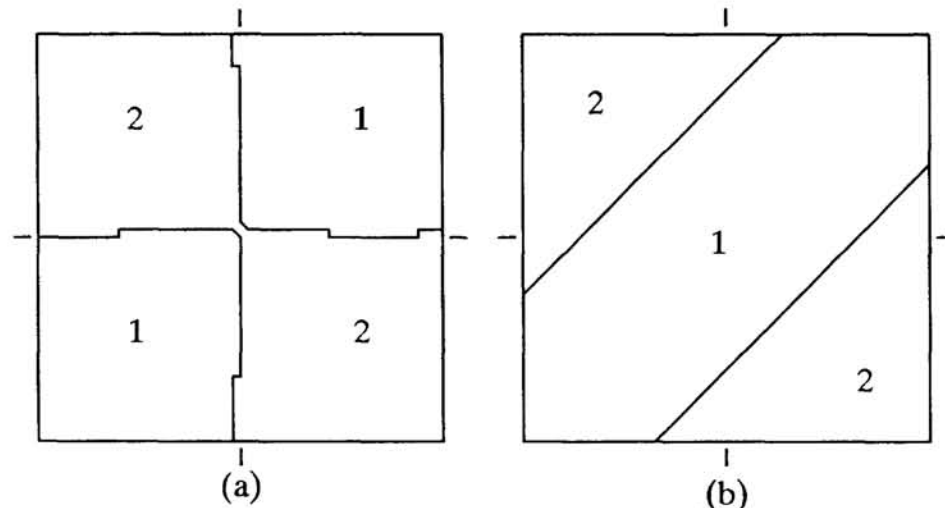

(a)                                   (b)

**Figure 5.** Typical solutions for XOR problem using (a) Kalman algorithm and (b) backprop.

fewer iterations. For the XOR problem, back-propagation failed to converge on any of the cases considered while the Kalman algorithm was able to find solutions with the same network configurations.

**References**

[1]    B. D. O. Anderson and J. B. Moore, *Optimal Filtering*, Prentice Hall, 1979.

[2]    A. Gelb, Ed., *Applied Optimal Estimation*, MIT Press, 1974.

[3]    B. Irie, and S. Miyake, "Capabilities of Three-layered Perceptrons," *Proceedings of the IEEE International Conference on Neural Networks*, San Diego, June 1988, Vol. I, pp. 641-648.

[4]    R. E. Kalman, "A New Approach to Linear Filtering and Prediction Problems," *J. Basic Eng., Trans. ASME*, Series D, Vol 82, No.1, 1960, pp. 35-45.

[5]    R. W. Prager and F. Fallside, "The Modified Kanerva Model for Automatic Speech Recognition," in *1988 IEEE Workshop on Speech Recognition*, Arden House, Harriman NY, May 31-June 3, 1988.

[6]    D. E. Rumelhart, G. E. Hinton and R. J. Williams, "Learning Internal Representations by Error Propagation," in D. E. Rumelhart and J. L. McCelland (Eds.), *Parallel Distributed Processing: Explorations in the Microstructure of Cognition. Vol 1: Foundations*. MIT Press, 1986.

[7]    A. Waibel, T. Hanazawa, G. Hinton, K. Shikano and K. Lang "Phoneme Recognition Using Time-Delay Neural Networks," *ATR internal Report* TR-I-0006, October 30, 1987.
